# Unification of Information Maximization and Minimization

**Ryotaro Kamimura**
Information Science Laboratory
Tokai University
1117 Kitakaname Hiratsuka Kanagawa 259-12, Japan
E-mail: ryo@cc.u-tokai.ac.jp

## Abstract

In the present paper, we propose a method to unify information maximization and minimization in hidden units. The information maximization and minimization are performed on two different levels: collective and individual level. Thus, two kinds of information: collective and individual information are defined. By maximizing collective information and by minimizing individual information, simple networks can be generated in terms of the number of connections and the number of hidden units. Obtained networks are expected to give better generalization and improved interpretation of internal representations. This method was applied to the inference of the maximum onset principle of an artificial language. In this problem, it was shown that the individual information minimization is not contradictory to the collective information maximization. In addition, experimental results confirmed improved generalization performance, because over-training can significantly be suppressed.

## 1 Introduction

There have been many attempts to interpret neural networks from the information theoretical point of view [2], [4], [5]. Applied to the supervised learning, information has been maximized and minimized, depending on problems. In these methods, information is defined by the outputs of hidden units. Thus, the methods aim to control hidden unit activity patterns in an optimal manner. Information maximization methods have been used to interpret explicitly internal representations and simultaneously to reduce the number of necessary hidden units [5]. On the other hand, information minimization methods have been especially used to improve generalization performance [2], [4] and to speed up learning. Thus, if it is possible to

maximize and minimize information simultaneously, information theoretic methods are expected to be applied to a wide range of problems.

In this paper, we unify the above mentioned two methods, namely, information maximization and minimization methods, into one framework to improve generalization performance and to interpret explicitly internal representations. However, it is apparently impossible to maximize and minimize simultaneously the information defined by the hidden unit activity. Our goal is to maximize and to minimize information on two different levels, namely, collective and individual levels. This means that information can be maximized in collective ways and information is minimized for individual input-hidden connections. The seeming contradictory proposition of the simultaneous information maximization and minimization can be overcome by assuming the existence of the two levels for the information control.

Information is supposed to be controlled by an information controller located outside neural networks and used exclusively to control information. By assuming the information controller, we can clearly see how information appropriately defined can be maximized or minimized. In addition, the actual implementation of information methods is much easier by introducing a concept of the information controller.

## 2    Concept of Information

In this section, we explain a concept of information in a general framework of an information theory. Let $Y$ take on a finite number of possible values $y_1, y_2, ..., y_M$ with probabilities $p(y_1), p(y_2), ..., p(y_M)$, respectively. Then, initial uncertainty $H(Y)$ of a random variable $Y$ is defined by

$$H(Y) = -\sum_{j=1}^{M} p(y_j) \log p(y_j). \tag{1}$$

Now, consider conditional uncertainty after the observation of another random variable $X$, taking possible values $x_1, x_2, ..., x_S$ with probabilities $p(x_1), p(x_2), ..., p(x_M)$, respectively. Conditional uncertainty $H(Y \mid X)$ can be defined as

$$H(Y \mid X) = -\sum_{s=1}^{S} p(x_s) \sum_{j=1}^{M} p(y_j \mid x_s) \log p(x_j \mid y_s). \tag{2}$$

We can easily verify that conditional uncertainty is always less than or equal to initial uncertainty. Information is usually defined as the decrease of this uncertainty [1].

$$
\begin{aligned}
I(Y \mid X) &= H(Y) - H(Y \mid X) \\
&= -\sum_{j=1}^{M} p(y_j) \log p(y_j) + \sum_{s=1}^{S} p(x_s) \sum_{j=1}^{M} p(y_j \mid x_s) \log p(y_j \mid x_s) \\
&= \sum_{s} \sum_{j} p(x_s) p(y_j \mid x_s) \log \frac{p(y_j \mid x_s)}{p(y_j)} \\
&= \sum_{s} p(x_s) I(Y \mid x_s)
\end{aligned}
\tag{3}
$$

where

$$I(Y \mid x_s) = \sum_{j} p(y_j \mid x_s) \log \frac{p(y_j \mid x_s)}{p(y_j)}$$

$$= -\sum_j p(y_j \mid x_s) \log p(y_j) + \sum_j p(y_j \mid x_s) \log p(y_j \mid x_s), \quad (4)$$

which is referred to as conditional information. Especially, when prior uncertainty is maximum, that is, a prior probability is equi-probable $(1/M)$, then information is

$$I(Y \mid X) = \log M + \sum_{s=1}^{S} p(x_s) \sum_{j=1}^{M} p(y_j \mid x_s) \log p(y_j \mid x_s) \quad (5)$$

where $\log M$ is maximum uncertainty concering $A$.

## 3   Formulation of Information Controller

In this section, we apply a concept of information to actual network architectures and define collective information and individual information. The notation in the above section is changed into ordinary notation used in the neural network.

### 3.1   Unification by Information Controller

Two kinds of information, collective information and individual information, are controlled by using an information controller. The information controller is devised to interpret the mechanism of the information maximization and minimization more explicitly. As shown in Figure 1, the information controller is composed of two subcomponents, that is, an individual information minimizer and collective information maximizer. A collective information maximizer is used to increase collective information as much as possible. An individual information minimizer is used to decrease individual information. By this minimization, the majority of connections are pushed toward zero. Eventually, all the hidden units tend to be intermediately activated. Thus, when the collective information maximizer and individual information maximizer are simultaneously applied, a hidden unit activity pattern is a pattern of the maximum information in which only one hidden unit is on, while all the other hidden units are off. However, multiple strongly negative connections to produce a maximum information state, are replaced by extremely weak input-hidden connections. Strongly negative connections are inhibited by the individual information minimization. This means that by the information controller, information can be maximized and at the same time one of the most important properties of the information minimization, namely, weight decay or weight elimination, can approximately be realized. Consequently, the information controller can generate much simplified networks in terms of hidden units and in terms of input-hidden connections.

### 3.2   Collective Information Maximizer

A neural network to be controlled is composed of input, hidden and output units with bias, as shown in Figure 1. The $j$th hidden unit receives a net input from input units and at the same time from a collective information maximizer:

$$u_j^s = x_j + \sum_{k=0}^{L} w_{jk} \xi_k^s \quad (6)$$

where $x_j$ is an information maximizer from the $j$th collective information maximizer to the $j$th hidden unit, $L$ is the number of input units, $w_{jk}$ is a connection from the $k$th input unit to the $j$th hidden unit and $\xi_k^s$ is the $k$th element of the $s$th input

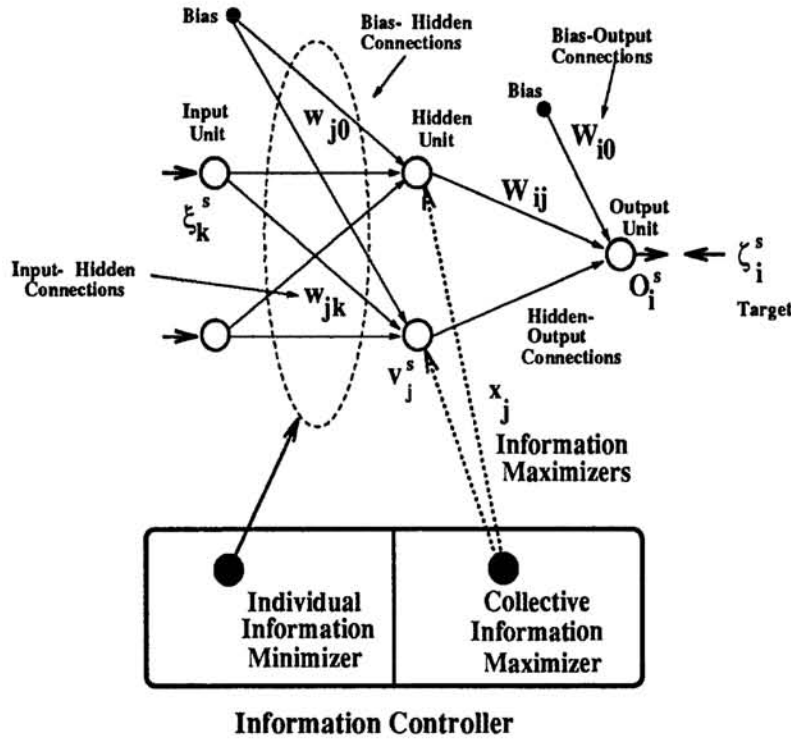

**Information Controller**

Figure 1: A network architecture, realizing the information controller.

pattern. The $j$th hidden unit produces an activity or an output by a sigmoidal activation function:

$$
\begin{aligned}
v_j^s &= f(u_j^s) \\
&= \frac{1}{1+\exp(-u_j^s)}.
\end{aligned}
\tag{7}
$$

The collective information maximizer is used to maximize the information contained in hidden units. For this purpose, we should define collective information. Now, suppose that in the previous formulation in information, a symbol $X$ and $Y$ represent a set of input patterns and hidden units respectively. Then, let us approximate a probability $p(y_j \mid x_s)$ by a normalized output $p_j^s$ of the $j$th hidden unit computed by

$$
p_j^s = \frac{v_j^s}{\sum_{m=1}^{M} v_m^s}
\tag{8}
$$

where the summation is over all the hidden units. Then, it is reasonable to suppose that at an initial stage all the hidden units are activated randomly or uniformly and all the input patterns are also randomly given to networks. Thus, a probability $p(y_j)$ of the activation of hidden units at the initial stage is equi-probable, that is, $1/M$. A probability $p(x_s)$ of input patterns is also supposed to be equi-probable, namely, $1/S$. Thus, information in the equation (3) is rewritten as

$$
I(Y \mid X) \approx -\sum_{j=1}^{M} \frac{1}{M} \log \frac{1}{M} + \frac{1}{S} \sum_{s=1}^{S} \sum_{j=1}^{M} p_j^s \log p_j^s
$$

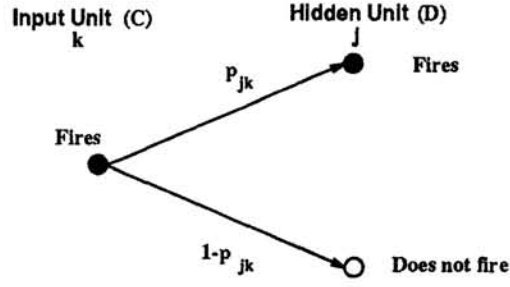

Figure 2: An interpretation of an input-hidden connection for defining the individual information.

$$= \log M + \frac{1}{S} \sum_{s=1}^{S} \sum_{j=1}^{M} p_j^s \log p_j^s \tag{9}$$

where $\log M$ is maximum uncertainty. This information is considered to be the information acquired in a course of learning. Information maximizers are updated to increase collective information. For obtaining update rules, we should differentiate the information function with respect to information maximizers $x_j$:

$$\Delta x_j = \beta S \frac{\partial I(Y \mid X)}{\partial x_j}$$

$$= \beta \sum_{s=1}^{S} \left( \log p_j^s - \sum_{m=1}^{M} p_m^s \log p_m^s \right) p_j^s (1 - v_j^s) \tag{10}$$

where $\beta$ is a parameter.

### 3.3  Individual Information Minimization

For representing individual information by a concept of information discussed in the previous section, we consider an output $p_{jk}$ from the $j$th hidden unit only with a connection from the $k$th input unit to the $j$th output unit:

$$p_{jk} = f(w_{jk}) \tag{11}$$

which is supposed to be a probability of the firing of the $j$th hidden unit, given the firing of the $k$th input unit, as shown in Figure 2. Since this probability is considered to be a probability, given the firing of the $k$th input unit, conditional information is appropriate for measuring the information. In addition, it is reasonable to suppose that a probability of the firing of the $j$th hidden unit is 1/2 at an initial stage of learning, because we have no knowledge on hidden units. Thus, conditional information for a pair of the $k$th unit and the $j$th hidden unit is formulated as

$$
\begin{aligned}
I_{jk}(D \mid fires) &\approx -p_{jk} \log \frac{1}{2} - (1 - p_{jk}) \log \left( 1 - \frac{1}{2} \right) \\
&\quad + p_{jk} \log p_{jk} + (1 - p_{jk}) \log(1 - p_{jk}) \\
&= \log 2 + p_{jk} \log p_{jk} + (1 - p_{jk}) \log(1 - p_{jk}) \tag{12}
\end{aligned}
$$

If connections are close to zero, this function is close to minimum information, meaning that it is impossible to estimate the firing of the $k$th hidden unit. If

Table 1: An example of obtained input-hidden connections $w_{jk}$ by the information controller. The parameter $\beta$, $\mu$ and $\eta$ were 0.015, 0.0008, and 0.01.

| Hidden | Input Units $\xi_k^s$ | | | | Bias $w_{j0}$ | Information |
|---|---|---|---|---|---|---|
| Units $v_j^s$ | 1 | 2 | 3 | 4 | | Maximizer $x_j$ |
| 1 | 3.09 | 10.77 | 26.48 | 13.82 | 22.07 | -60.88 |
| 2 | -3.35 | 0.11 | 0.33 | -3.08 | -0.95 | 1.63 |
| 3 | -0.01 | 0.00 | 0.00 | -0.01 | 0.00 | -10.93 |
| 4 | 0.00 | 0.00 | 0.00 | 0.00 | 0.00 | -10.94 |
| 5 | 0.00 | 0.00 | -0.01 | 0.00 | 0.00 | -10.97 |
| 6 | 0.02 | 0.01 | -0.04 | 0.01 | 0.06 | -12.01 |
| 7 | 0.00 | 0.00 | -0.01 | 0.00 | 0.00 | -11.01 |
| 8 | 0.00 | 0.00 | -0.01 | 0.00 | 0.00 | -11.00 |
| 9 | 0.02 | 0.01 | -0.03 | 0.01 | 0.03 | -11.61 |
| 10 | 0.01 | 0.00 | -0.02 | 0.00 | 0.07 | -11.67 |

connections are larger, the information is larger and correlation between input and hidden units is larger. Total individual information is the sum of all the individual individual information, namely,

$$I(D \mid fires) \;=\; \sum_{j=1}^{M}\sum_{k=0}^{L} I_{jk}(D \mid fires), \tag{13}$$

because each connection is treated separately or independently. The individual information minimization directly controls the input-hidden connections. By differentiating the individual information function and a cross entropy cost function with respect to input-hidden connections $w_{jk}$, we have rules for updating concerning input-hidden connections:

$$\begin{aligned}
\Delta w_{jk} &= -\mu \frac{\partial I(D \mid fires)}{\partial w_{jk}} - \eta \frac{\partial G}{\partial w_{jk}} \\
&= -\mu \, w_{jk} \, p_{jk}(1 - p_{jk}) + \eta \sum_{s=1}^{S} \delta_j^s \xi_k^s
\end{aligned} \tag{14}$$

where $\delta_j^s$ is an ordinary delta for the cross entropy function and $\eta$ and $\mu$ are parameters. Thus, rules for updating with respect to input-hidden connections are closely related to the weight decay method. Clearly, as the individual information minimization corresponds to diminishing the strength of input-hidden connections.

## 4    Results and Discussion

The information controller was applied to the segmentation of strings of an artificial language into appropriate minimal elements, that is, syllables. Table 1 shows input-hidden connections with the bias and the information maximizers. Hidden units were ordered by the magnitude of the relevance of each hidden unit [6]. Collective information and individual information could sufficiently be maximized and minimized. Relative collective and individual information were 0.94 and 0.13. In this state, all the input-hidden connections except connections into the first two hidden units are almost zero. Information maximizers $x_j$ are all strongly negative for these cases. These negative information maximizers make eight hidden units (from the third to tenth hidden unit) inactive, that is, close to zero. By carefully

Table 2: Generalization performance comparison for 200 and 200 training patterns. Averages in the table are average generalization errors over seven errors of ten errors with ten different initial values.

(a) 200 patterns

| Methods | Generalization Errors | | | |
| | RMS | | Error Rates | |
| | Averages | Std. Dev. | Averages | Std. Dev. |
|---|---|---|---|---|
| Standard | 0.188 | 0.010 | 0.087 | 0.015 |
| Weight Decay | 0.183 | 0.004 | 0.082 | 0.009 |
| Weight Elimination | 0.172 | 0.014 | 0.064 | 0.015 |
| Information Controller | 0.167 | 0.011 | 0.052 | 0.008 |

(b) 300 patterns

| Methods | Generalization Errors | | | |
| | RMS | | Error Rates | |
| | Averages | Std. Dev. | Averages | Std. Dev. |
|---|---|---|---|---|
| Standard | 0.108 | 0.009 | 0.024 | 0.009 |
| Weight Decay | 0.110 | 0.003 | 0.012 | 0.004 |
| Weight Elimination | 0.083 | 0.005 | 0.009 | 0.006 |
| Information Controller | 0.072 | 0.006 | 0.008 | 0.004 |

examing the first two hidden units, we could see that the first hidden unit and the second hidden unit are concerned with rules for syllabification and a exceptional case.

Then, networks were trained to infer the well-formedness of strings in addition to the segmentation to examine generalization performance. Table 2 shows generalization errors for 200 and 300 training patterns. As clearly shown in the figure, the best generalization performance in terms of *RMS* and error rates is obtained by the information controller. Thus, experimental results confirmed that in all cases the generalization performance of the information controller is well over the other methods. In addition, experimental results explicitly confirmed that better generalization performance is due to the suppression of over-training by the information controller.

# References

[1] R. Ash, *Information Theory*, John Wiley & Sons: New York, 1965.

[2] G. Deco, W. Finnof and H. G. Zimmermann, "Unsupervised mutual information criterion for elimination of overtraining in Supervised Multilayer Networks," *Neural Computation*, Vol. 7, pp.86-107, 1995.

[3] R. Kamimura "Entropy minimization to increase the selectivity: selection and competition in neural networks," *Intelligent Engineering Systems through Artificial Neural Networks*, ASME Press, pp.227-232, 1992.

[4] R. Kamimura, T. Takagi and S. Nakanishi, "Improving generalization performance by information minimization," *IEICE Transactions on Information and Systems*, Vol. E78-D, No.2, pp.163-173, 1995.

[5] R. Kamimura and S. Nakanishi, "Hidden information maximization for feature detection and rule discovery," *Network: Computation in Neural Systems*, Vol.6, pp.577-602, 1995.

[6] M. C. Mozer and P. Smolensky, "Using relevance to reduce network size automatically," *Connection Science*, Vo.1, No.1, pp.3-16, 1989.
